# Recovering Intrinsic Images from a Single Image

**Marshall F Tappen**     **William T Freeman**     **Edward H Adelson**
MIT Artificial Intelligence Laboratory
Cambridge, MA 02139
*mtappen@ai.mit.edu, wtf@ai.mit.edu, adelson@ai.mit.edu*

## Abstract

We present an algorithm that uses multiple cues to recover shading and reflectance intrinsic images from a single image. Using both color information and a classifier trained to recognize gray-scale patterns, each image derivative is classified as being caused by shading or a change in the surface's reflectance. Generalized Belief Propagation is then used to propagate information from areas where the correct classification is clear to areas where it is ambiguous. We also show results on real images.

## 1   Introduction

Every image is the product of the characteristics of a scene. Two of the most important characteristics of the scene are its shading and reflectance. The shading of a scene is the interaction of the surfaces in the scene and the illumination. The reflectance of the scene describes how each point reflects light. The ability to find the reflectance of each point in the scene and how it is shaded is important because interpreting an image requires the ability to decide how these two factors affect the image. For example, the geometry of an object in the scene cannot be recovered without being able to isolate the shading of every point. Likewise, segmentation would be simpler given the reflectance of each point in the scene. In this work, we present a system which finds the shading and reflectance of each point in a scene by decomposing an input image into two images, one containing the shading of each point in the scene and another image containing the reflectance of each point. These two images are types of a representation known as *intrinsic images* [1] because each image contains one intrinsic characteristic of the scene.

Most prior algorithms for finding shading and reflectance images can be broadly classified as generative or discriminative approaches. The generative approaches create possible surfaces and reflectance patterns that explain the image, then use a model to choose the most likely surface. Previous generative approaches include modeling worlds of painted polyhedra [11] or constructing surfaces from patches taken out of a training set [3]. In contrast, discriminative approaches attempt to differentiate between changes in the image caused by shading and those caused by a reflectance change. Early algorithms, such as Retinex [8], were based on simple assumptions, such as the assumption that the gradients along reflectance changes have much larger magnitudes than those caused by shading. That assumption does not hold for many real images, so recent algorithms have used more complex statistics to separate shading and reflectance. Bell and Freeman [2] trained a classifier to use local image information to classify steerable pyramid coefficients as being due to

shading or reflectance. Using steerable pyramid coefficients allowed the algorithm to classify edges at multiple orientations and scales. However, the steerable pyramid decomposition has a low-frequency residual component that cannot be classified. Without classifying the low-frequency residual, only band-pass filtered copies of the shading and reflectance images can be recovered. In addition, low-frequency coefficients may not have a natural classification.

In a different direction, Weiss [13] proposed using multiple images where the reflectance is constant, but the illumination changes. This approach was able to create full frequency images, but required multiple input images of a fixed scene.

In this work, we present a system which uses multiple cues to recover full-frequency shading and reflectance intrinsic images from a single image. Our approach is discriminative, using both a classifier based on color information in the image and a classifier trained to recognize local image patterns to distinguish derivatives caused by reflectance changes from derivatives caused by shading. We also address the problem of ambiguous local evidence by using a Markov Random Field to propagate the classifications of those areas where the evidence is clear into ambiguous areas of the image.

## 2  Separating Shading and Reflectance

Our algorithm decomposes an image into shading and reflectance images by classifying each image derivative as being caused by shading or a reflectance change. We assume that the input image, $\mathcal{I}(x, y)$, can be expressed as the product of the shading image, $\mathcal{S}(x, y)$, and the reflectance image, $\mathcal{R}(x, y)$. Considering the images in the log domain, the derivatives of the input image are the sum of the derivatives of the shading and reflectance images. It is unlikely that significant shading boundaries and reflectance edges occur at the same point, thus we make the simplifying assumption that every image derivative is either caused by shading or reflectance. This reduces the problem of specifying the shading and reflectance derivatives to that of binary classification of the image's $x$ and $y$ derivatives.

Labelling each $x$ and $y$ derivative produces estimates of the derivatives of the shading and reflectance images. Each derivative represents a set of linear constraints on the image and using both derivative images results in an over-constrained system. We recover each intrinsic image from its derivatives by using the method introduced by Weiss in [13] to find the pseudo-inverse of the over-constrained system of derivatives. If $f_x$ and $f_y$ are the filters used to compute the $x$ and $y$ derivatives and $\mathcal{F}_x$ and $\mathcal{F}_y$ are the estimated derivatives of shading image, then the shading image, $\mathcal{S}(x, y)$ is:

$$\mathcal{S}(x, y) = g \star [(f_x(-x, -y) \star \mathcal{F}_x) + (f_y(-x, -y) \star \mathcal{F}_y)] \tag{1}$$

where $\star$ is convolution, $f(-x, -y)$ is a reversed copy of $f(x, y)$, and $g$ is the solution of

$$g \star [(f_x(-x, -y) \star f_x(x, y)) + (f_y(-x, -y) \star f_x(x, y))] = \delta \tag{2}$$

The reflectance image is found in the same fashion. One nice property of this technique is that the computation can be done using the FFT, making it more computationally efficient.

## 3  Classifying Derivatives

With an architecture for recovering intrinsic images, the next step is to create the classifiers to separate the underlying processes in the image. Our system uses two classifiers, one which uses color information to separate shading and reflectance derivatives and a second classifier that uses local image patterns to classify each derivative.

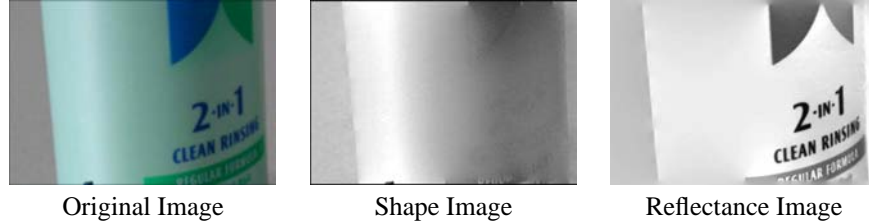

| Original Image | Shape Image | Reflectance Image |

Figure 1: Example computed using only color information to classify derivatives. To facilitate printing, the intrinsic images have been computed from a gray-scale version of the image. The color information is used solely for classifying derivatives in the gray-scale copy of the image.

### 3.1 Using Color Information

Our system takes advantage of the property that changes in color between pixels indicate a reflectance change [10]. When surfaces are diffuse, any changes in a color image due to shading should affect all three color channels proportionally. Assume two adjacent pixels in the image have values $c_1$ and $c_2$, where $c_1$ and $c_2$ are RGB triplets. If the change between the two pixels is caused by shading, then only the intensity of the color changes and $c_2 = \alpha c_1$ for some scalar $\alpha$. If $c_2 \neq \alpha c_1$, the chromaticity of the colors has changed and the color change must have been caused by a reflectance change. A chromaticity change in the image indicates that the reflectance must have changed at that point.

To find chromaticity changes, we treat each RGB triplet as a vector and normalize them to create $\hat{c}_1$ and $\hat{c}_2$. We then use the angle between $\hat{c}_1$ and $\hat{c}_2$ to find reflectance changes. When the change is caused by shading, $(\hat{c}_1 \cdot \hat{c}_2)$ equals 1. If $(\hat{c}_1 \cdot \hat{c}_2)$ is below a threshold, then the derivative associated with the two colors is classified as a reflectance derivative. Using only the color information, this approach is similar to that used in [6]. The primary difference is that our system classifies the vertical and horizontal derivatives independently.

Figure 1 shows an example of the results produced by the algorithm. The classifier marked all of the reflectance areas correctly and the text is cleanly removed from the bottle. This example also demonstrates the high quality reconstructions that can be obtained by classifying derivatives.

### 3.2 Using Gray-Scale Information

While color information is useful, it is not sufficient to properly decompose images. A change in color intensity could be caused by either shading or a reflectance change. Using only local color information, color intensity changes cannot be classified properly. Fortunately, shading patterns have a unique appearance which can be discriminated from most common reflectance patterns. This allows us to use the local gray-scale image pattern surrounding a derivative to classify it.

The basic feature of the gray-scale classifier is the absolute value of the response of a linear filter. We refer to a feature computed in this manner as a *non-linear filter*. The output of a non-linear, $F$, given an input patch $I_p$ is

$$F = |I_p \star w| \tag{3}$$

where $\star$ is convolution and $w$ is a linear filter. The filter, $w$ is the same size as the image patch, $I$, and we only consider the response at the center of $I_p$. This makes the feature a function from a patch of image data to a scalar response. This feature could also be viewed as the absolute value of the dot product of $I_p$ and $w$. We use the responses of linear

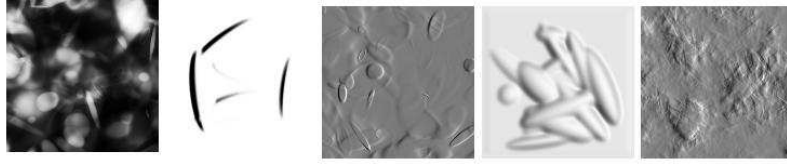

Figure 2: Example images from the training set. The first two are examples of reflectance changes and the last three are examples of shading

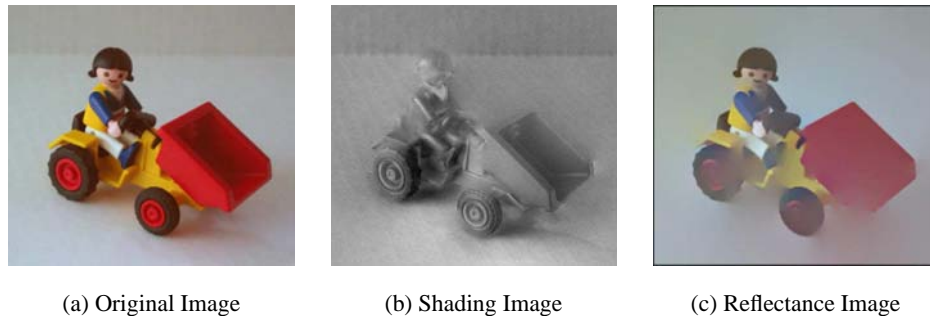

(a) Original Image          (b) Shading Image          (c) Reflectance Image

Figure 3: Results obtained using the gray-scale classifier.

filters as the basis for our feature, in part, because they have been used successfully for characterizing [9] and synthesizing [7] images of textured surfaces.

The non-linear filters are used to classify derivatives with a classifier similar to that used by Tieu and Viola in [12]. This classifier uses the AdaBoost [4] algorithm to combine a set of weak classifiers into a single strong classifier. Each weak classifier is a threshold test on the output of one non-linear filter. At each iteration of the AdaBoost algorithm, a new weak classifier is chosen by choosing a non-linear filter and a threshold. The filter and threshold are chosen greedily by finding the combination that performs best on the re-weighted training set. The linear filter in each non-linear filter is chosen from a set of oriented first and second derivative of Gaussian filters.

The training set consists of a mix of images of rendered fractal surfaces and images of shaded ellipses placed randomly in the image. Examples of reflectance changes were created using images of random lines and images of random ellipse painted onto the image. Samples from the training set are shown in 2. In the training set, the illumination is always coming from the right side of the image. When evaluating test images, the classifier will assume that the test image is also lit from the right.

Figure 3 shows the results of our system using only the gray-scale classifier. The results can be evaluated by thinking of the shading image as how the scene should appear if it were made entirely of gray plastic. The reflectance image should appear very flat, with the the three-dimensional depth cues placed in the shading image. Our system performs well on the image shown in Figure 3. The shading image has a very uniform appearance, with almost all of the effects of the reflectance changes placed in the reflectance image.

The examples shown are computed without taking the log of the input image before processing it. The input images are uncalibrated and ordinary photographic tonescale is very similar to a log transformation. Errors from not taking log of the input image first would

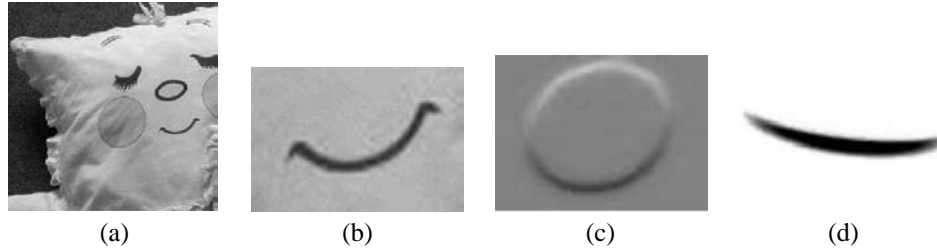

(a)　　　　　　　(b)　　　　　　　(c)　　　　　　　(d)

Figure 4: An example where propagation is needed. The smile from the pillow image in (a) has been enlarged in (b). Figures (c) and (d) contain an example of shading and a reflectance change, respectively. Locally, the center of the mouth in (b) is as similar to the shading example in (c) as it is to the example reflectance change in (d).

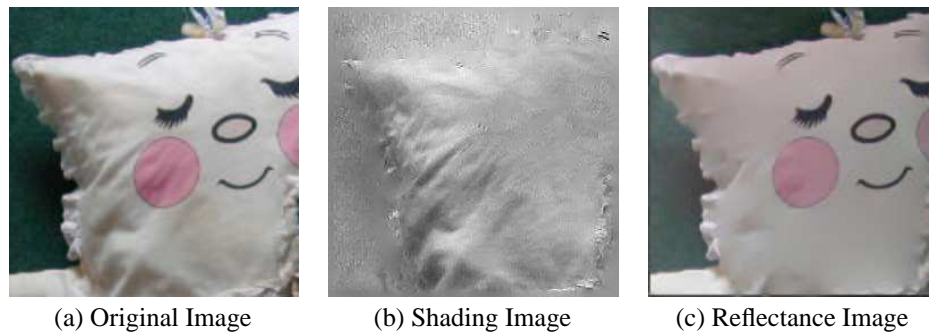

(a) Original Image　　　(b) Shading Image　　　(c) Reflectance Image

Figure 5: The pillow from Figure 4. This is found by combining the local evidence from the color and gray-scale classifiers, then using Generalized Belief Propagation to propagate local evidence.

cause one intrinsic image to modulate the local brightness of the other. However, this does not occur in the results.

## 4  Propagating Evidence

While the classifier works well, there are still areas in the image where the local information is ambiguous. An example of this is shown in Figure 4. When compared to the example shading and reflectance change in Figure 4(c) and 4(d), the center of the mouth in Figure 4(b) is equally well classified with either label. However, the corners of the mouth can be classified as being caused by a reflectance change with little ambiguity. Since the derivatives in the corner of the mouth and the center all lie on the same image contour, they should have the same classification. A mechanism is needed to propagate information from the corners of the mouth, where the classification is clear, into areas where the local evidence is ambiguous. This will allow areas where the classification is clear to disambiguate those areas where it is not.

In order to propagate evidence, we treat each derivative as a node in a Markov Random Field with two possible states, indicating whether the derivative is caused by shading or caused by a reflectance change. Setting the compatibility functions between nodes correctly will force nodes along the same contour to have the same classification.

## 4.1 Model for the Potential Functions

Each node in the MRF corresponds to the classification of a derivative. We constrain the compatibility functions for two neighboring nodes, $x_i$ and $x_j$, to be of the form

$$\psi(x_i, x_j) = \left[ \begin{array}{cc} \beta & 1 - \beta \\ 1 - \beta & \beta \end{array} \right] \tag{4}$$

with $0 \leq \beta \leq 1$.

The term $\beta$ controls how much the two nodes should influence each other. Since derivatives along an image contour should have the same classification, $\beta$ should be close to 1 when two neighboring derivatives are along a contour and should be 0.5 when no contour is present.

Since $\beta$ depends on the image at each point, we express it as $\beta(\mathcal{I}_{xy})$, where $\mathcal{I}_{xy}$ is the image information at some point. To ensure $\beta(\mathcal{I}_{xy})$ between 0 and 1, it is modelled as $\beta(\mathcal{I}_{xy}) = g(z(\mathcal{I}_{xy}))$, where $g(\cdot)$ is the logistic function and $z(\mathcal{I}_{xy})$ has a large response along image contours.

## 4.2 Learning the Potential Functions

The function $z(\mathcal{I}_{xy})$ is based on two local image features, the magnitude of the image and the difference in orientation between the gradient and the orientation of the graph edge. These features reflect our heuristic that derivatives along an image contour should have the same classification.

The difference in orientation between a horizontal graph edge and image contour, $\hat{\phi}$, is found from the orientation of the image gradient, $\phi$. Assuming that $-\pi/2 \leq \phi \leq \pi/2$, the angle between a horizontal edge and the image gradient, $\hat{\phi}$, is $\hat{\phi} = |\phi|$. For vertical edges, $\hat{\phi} = |\phi| - \pi/2$.

To find the values of $z(\cdot)$ we maximize the probability of a set of the training examples over the parameters of $z(\cdot)$. The examples are taken from the same set used to train the gray-scale classifiers. The probability of training samples is

$$P = \frac{1}{Z} \prod_{(i,j)} \psi(x_i, x_j) \tag{5}$$

where all $(i, j)$ are the indices of neighboring nodes in the MRF and $Z$ is a normalization constant. Note that each $\psi(\cdot)$ is a function of $z(\mathcal{I}_{xy})$.

The function relating the image features to $\psi(\cdot)$, $z(\cdot)$, is chosen to be a linear function and is found by maximizing equation 5 over a set of training images similar to those used to train the local classifier. In order to simplify the training process, we approximate the true probability in Equation 5 by assuming that $Z$ is constant. Doing so leads to the following value of $z(\cdot)$:

$$z(\hat{\phi}, |\nabla \mathcal{I}|) = -1.2 \times \hat{\phi} + 1.62 \times |\nabla \mathcal{I}| + 2.3 \tag{6}$$

where $|\nabla \mathcal{I}|$ is the magnitude of the image gradient and both $\hat{\phi}$ and $|\nabla \mathcal{I}|$ have been normalized to be between 0 and 1. These measures break down in areas with a weak gradient, so we set $\beta(\mathcal{I}_{xy})$ to 0.5 for regions of the image with a gradient magnitude less than 0.05. Combined with the values learned for $z(\cdot)$, this effectively limits $\beta$ to the range $0.5 \leq \beta \leq 1$.

Larger values of $z(\cdot)$ correspond to a belief that the derivatives connected by the edge should have the same value, while negative values signify that the derivatives should have

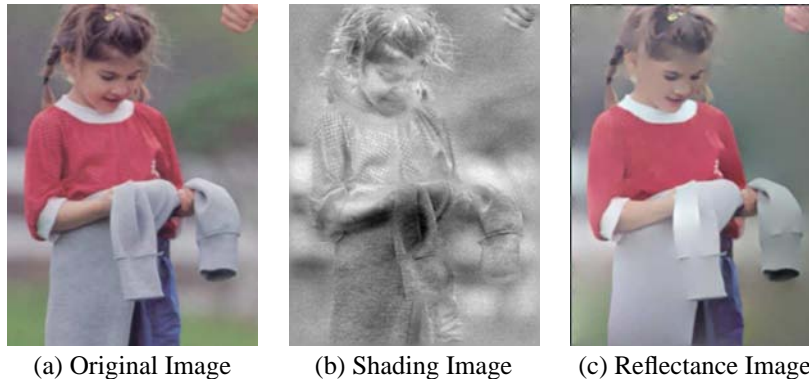

| (a) Original Image | (b) Shading Image | (c) Reflectance Image |

Figure 6: Example generated by combining color and gray-scale information, along with using propagation.

a different value. The values in equation 6 correspond with our expected results; two derivatives are constrained to have the same value when they are along an edge in the image that has a similar orientation to the edge in the MRF connecting the two nodes.

### 4.3    Inferring the Correct Labelling

Once the compatibility functions have been learned, the label of each derivative can be inferred. The local evidence for each node in the MRF is obtained from the results of the color classifier and from the gray-scale classifier by assuming that the two are statistically independent. It is necessary to use the color information because propagation cannot help in areas where the gray-scale classifier misses an edge altogether. In Figure 5, the cheek patches on the pillow, which are pink in the color image, are missed by the gray-scale classifier, but caught by the color classifier. For the results shown, we used the results of the AdaBoost classifier to classify the gray-scale images and used the method suggested by Friedman et al. to obtain the probability of the labels [5].

We used the Generalized Belief Propagation algorithm [14] to infer the best label of each node in the MRF because ordinary Belief Propagation performed poorly in areas with both weak local evidence and strong compatibility constraints. The results of using color, gray-scale information, and propagation can be seen in Figure 5. The ripples on the pillow are correctly identified as being caused by shading, while the face is correctly identified as having been painted on. In a second example, shown in Figure 6, the algorithm correctly identifies the change in reflectance between the sweatshirt and the jersey and correctly identifies the folds in the clothing as being caused by shading. There are some small shading artifacts in the reflectance image, especially around the sleeves of the sweatshirt, presumably caused by particular shapes not present in the training set. All of the examples were computed using ten non-linear filters as input for the AdaBoost gray-scale classifier.

## 5    Discussion

We have presented a system that is able to use multiple cues to produce shading and reflectance intrinsic images from a single image. This method is also able to produce satisfying results for real images. The most computationally intense steps for recovering the shading and reflectance images are computing the local evidence, which takes about six minutes on a 700MHz Pentium for a $256 \times 256$ image, and running the Generalized Belief Propagation algorithm. Belief propagation was used on both the $x$ and $y$ derivative images

and took around 6 minutes to run 200 iterations on each image. The pseudo-inverse process took under 5 seconds.

The primary limitation of this method lies in the classifiers. For each type of surface, the classifiers must incorporate knowledge about the structure of the surface and how it appears when illuminated. The present classifiers operate at a single spatial scale, however the MRF framework allows the integration of information from multiple scales.

## Acknowledgments

Portions of this work were completed while W.T.F was a Senior Research Scientist and M.F.T was a summer intern at Mitsubishi Electric Research Labs. This work was supported by an NDSEG fellowship to M.F.T, by NIH Grant EY11005-04 to E.H.A., by a grant from NTT to E.H.A., and by a contract with Unilever Research.

## References

[1] H. G. Barrow and J. M. Tenenbaum. Recovering intrinsic scene characteristics from images. In *Computer Vision Systems*, pages 3–26. Academic Press, 1978.

[2] M. Bell and W. T. Freeman. Learning local evidence for shading and reflection. In *Proceedings International Conference on Computer Vision*, 2001.

[3] W. T. Freeman, E. C. Pasztor, and O. T. Carmichael. Learning low-level vision. *International Journal of Computer Vision*, 40(1):25–47, 2000.

[4] Y. Freund and R. E. Schapire. A decision-theoretic generalization of on-line learning and an application to boosting. *Journal of Computer and System Sciences*, 55(1):119–139, 1997.

[5] J. Friedman, T. Hastie, and R. Tibshirami. Additive logistic regression: A statistical view of boosting. *The Annals of Statistics*, 38(2):337–374, 2000.

[6] B. V. Funt, M. S. Drew, and M. Brockington. Recovering shading from color images. In G. Sandini, editor, *ECCV-92: Second European Conference on Computer Vision*, pages 124–132. Springer-Verlag, May 1992.

[7] D. Heeger and J. Bergen. Pyramid-based texture analysis/synthesis. In *Computer Graphics Proceeding, SIGGRAPH 95*, pages 229–238, August 1995.

[8] E. H. Land and J. J. McCann. Lightness and retinex theory. *Journal of the Optical Society of America*, 61:1–11, 1971.

[9] T. Leung and J. Malik. Recognizing surfaces using three-dimensional textons. In *IEEE International Conference on Computer Vision*, 1999.

[10] J. M. Rubin and W. A. Richards. Color vision and image intensities: When are changes material. *Biological Cybernetics*, 45:215–226, 1982.

[11] P. Sinha and E. H. Adelson. Recovering reflectance in a world of painted polyhedra. In *Fourth International Conference on Computer Vision*, pages 156–163. IEEE, 1993.

[12] K. Tieu and P. Viola. Boosting image retrieval. In *Proceedings IEEE Computer Vision and Pattern Recognition*, volume 1, pages 228–235, 2000.

[13] Y. Weiss. Deriving intrinsic images from image sequences. In *Proceedings International Conference on Computer Vision*, Vancouver, Canada, 2001. IEEE.

[14] J. Yedidia, W. T. Freeman, and Y. Weiss. Generalized belief propagation. In *Advances in Neural Information Processing Systems 13*, pages 689–695, 2001.
